# A Unified Learning Scheme: Bayesian-Kullback Ying-Yang Machine

**Lei Xu**
1. Computer Science Dept., The Chinese University of HK, Hong Kong
2. National Machine Perception Lab, Peking University, Beijing

## Abstract

A Bayesian-Kullback learning scheme, called Ying-Yang Machine, is proposed based on the two complement but equivalent Bayesian representations for joint density and their Kullback divergence. Not only the scheme unifies existing major supervised and unsupervised learnings, including the classical maximum likelihood or least square learning, the maximum information preservation, the EM & *em* algorithm and information geometry, the recent popular Helmholtz machine, as well as other learning methods with new variants and new results; but also the scheme provides a number of new learning models.

## 1 INTRODUCTION

Many different learning models have been developed in the literature. We may come to an age of searching a unified scheme for them. With a unified scheme, we may understand deeply the existing models and their relationships, which may cause cross-fertilization on them to obtain new results and variants; We may also be guided to develop new learning models, after we get better understanding on which cases we have already studied or missed, which deserve to be further explored.

Recently, a Baysian-Kullback scheme, called the YING-YANG Machine, has been proposed as such an effort(Xu, 1995a). It bases on the Kullback divergence and two complement but equivalent Baysian representations for the joint distribution of the input space and the representation space, instead of merely using Kullback divergence for matching un-structuralized joint densities in information geometry type learnings (Amari, 1995a&b; Byrne, 1992; Csiszar, 1975). The two representations consist of four different components. The different combinations of choices of each component lead the YING-YANG Machine into different learning models. Thus, it acts as a general learning scheme for unifying the existing major unsupervised and supervised learnings. As shown in Xu(1995a), its one special case reduces to the EM algorithm (Dempster et al, 1977; Hathaway, 1986; Neal & Hinton, 1993)

and the closely related *Information Geometry* theory and the *em* algorithm (Amari, 1995a&b), to MDL autoencoder with a "bits-back" argument by Hinton & Zemel (1994) and its alternative equivalent form that minimizes the bits of uncoded residual errors and the unused bits in the transmission channel's capacity (Xu, 1995d), as well as to *Multisets modeling* learning (Xu, 1995e)–a unified learning framework for clustering, PCA-type learnings and self-organizing map. It other special case reduces to maximum information preservation (Linsker, 1989; Atick & Redlich, 1990; Bell & Sejnowski, 1995). More interestingly its another special case reduces to Helmholtz machine (Dayan et al,1995; Hinton, 1995) with new understandings. Moreover, the YING-YANG machine includes also maximum likelihood or least square learning.

Furthermore, the *YING-YANG Machine* has also been extended to temporal patterns with a number of new models for signal modeling. Some of them are the extensions of Helmholtz machine or maximum information preservation learning to temporal processing. Some of them include and extend the Hidden Markov Model (HMM), AMAR and AR models (Xu, 1995b). In addition, it has also been shown in Xu(1995a&c, 1996a) that one special case of the YING-YANG machine can provide us three variants for clustering or VQ, particularly with criteria and an automatic procedure developed for solving how to select the number of clusters in clustering analysis or Gaussian mixtures — a classical problem that remains open for decades.

In this paper, we present a deep and systematical further study. Section 2 redescribes the unified scheme on a more precise and systematical basis via discussing the possible marital status of the two Bayesian representations for joint density. Section 3 summarizes and explains those existing models under the unified scheme, particularly we have clarified some confusion made in the previous papers (Xu, 1995a&b) on maximum information preservation learning. Section 4 proposed and summarizes a number of possible new models suggested by the unified scheme.

## 2  BAYESIAN-KULLBACK YING-YANG MACHINE

As argued in Xu (1995a), unsupervised and supervised learning problems can be summarized into the problem of estimating joint density $P(x, y)$ of patterns in the input space $X$ and the representation space $Y$, as shown in Fig.1. Under the Bayesian framework, we have two representations for $P(x, y)$. One is $P_{M_1}(x, y) = P_{M_1}(y|x)P_{M_1}(x)$, implemented by a model $M_1$ called *YANG*/(male) part since it performs the task of transferring a pattern/(a real body) into a code/(a seed). The other is $P_{M_2}(x, y) = P_{M_2}(x|y)P_{M_2}(y)$, implemented by a model $M_2$ called *YING* part since it performs the task of generating a pattern/(a real body) from a code/(a seed). They are complement to each other and together implement an entire circle $x \rightarrow y \rightarrow x$. This compliments to the ancient chinese YING-YANG philosophy.

Here we have four components $P_{M_1}(x)$, $P_{M_1}(y|x)$, $P_{M_2}(x|y)$ and $P_{M_2}(y)$. The $P_{M_1}(x)$ can be fixed at some density estimate on input data, e.g., we have at least two choices–Parzen window estimate $P_h(x)$ or empirical estimate $P_0(x)$:

$$P_h(x) = \frac{1}{Nh^d} \sum_{i=1}^{N} K(\frac{x-x_i}{h}), \quad P_0(x) = \lim_{h\rightarrow 0} P_h(x) = \frac{1}{N} \sum_{i=1}^{N} \delta(x - x_i). \quad (1)$$

For $P_{M_1}(y|x)$, $P_{M_2}(x|y)$, each can have three choices: (1) from a *parametric* family specified by model $M_1$ or $M_2$; (2) free of model with $P_{M_1}(y|x) = P(y|x)$ or $P_{M_2}(x|y) = P(x|y)$; (3) broken channel $P_{M_1}(y|x) = P_{M_1}(y)$ or $P_{M_2}(x|y) = P_{M_2}(x)$. Finally, $P_{M_2}(y)$ with its $y$ consistent to $P_{M_1}(y|x)$ can also being from a *parametric* family or free of model. Any combinations of the choices of the four components forms a potential YING-YANG pair. We at least have $2 \times 3 \times 3 \times 2 = 36$ pairs.

A YING-YANG pair has four types of marital status: (a) *marry*, i.e., YING and

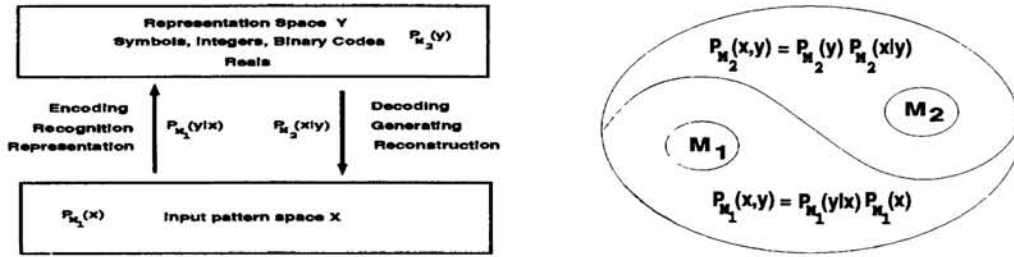

Figure 1   The joint spaces $X, Y$ and the YING-YANG Machine

YANG match each other; (b) *divorce*, i.e., YING and YANG go away from each other; (c) YING chases YANG, YANG escapes; (d) YANG chases YING, but YING escapes. The four types can be described by a combination of minimization (chasing) and maximization (escaping) on one of the two Kullback divergences below:

$$K(M_1, M_2) = \int_{x,y} P_{M_1}(y|x) P_{M_1}(x) \log \frac{P_{M_1}(y|x) P_{M_1}(x)}{P_{M_2}(x|y) P_{M_2}(y)} dxdy \quad (2a)$$

$$K(M_2, M_1) = \int_{x,y} P_{M_2}(x|y) P_{M_2}(y) \log \frac{P_{M_2}(x|y) P_{M_2}(y)}{P_{M_1}(y|x) P_{M_1}(x)} dxdy \quad (2b)$$

Table 1 Mathematical description for marital status of a YING-YANG pair

| $\min_{M_1,M_2} K(M_1,M_2)$ (a) | $\max_{M_1} \min_{M_2} K(M_1,M_2)$ | $\max_{M_2} \min_{M_1} K(M_1,M_2)$ |
|---|---|---|
| $\max_{M_1,M_2} K(M_1,M_2)$ (b) | $\min_{M_2} \max_{M_1} K(M_1,M_2)$ | $\min_{M_1} \max_{M_2} K(M_1,M_2)$ |

We can replace $K(M_1, M_2)$ by $K(M_2, M_1)$ in the table. The 2nd & 3rd columns are for (c) (d) respectively, each has two cases depending on who starts the act and the two are usually not equivalent. Their results are undefined depending on initial condition for $M_1, M_2$, except of two special cases: (i) Free $P_{M_1}(y|x)$ and parametric $P_{M_2}(x|y)$, with $\min_{M_2} \max_{M_1} K$ being the same as (b) with broken $P_{M_1}(y|x)$, and with $\max_{M_2} \min_{M_1} K$ defined but useless. (ii) Free $P_{M_2}(x|y)$ and parametric $P_{M_1}(y|x)$, with $\min_{M_1} \max_{M_2} K$ the same as case (a) with broken $P_{M_2}(x|y)$, with $\min_{M_1} \max_{M_2} K$ defined but useless.

Therefore, we will focus on the status *marry* and *divorce*. Even so, not all of the above mentioned $2 \times 3 \times 3 \times 2 = 36$ YING-YANG pairs provide sensible learning models although $\min_{M_1,M_2} K$ and $\max_{M_1,M_2} K$ are always well defined. Fortunately, a quite number of them indeed lead us to useful learning models, as will be shown in the sequent sections.

We can implement $\min_{M_1,M_2} K(M_1, M_2)$ by the following *Alternative Minimization* (ALTMIN) procedure:
**Step 1**   Fix $M_2 = M_2^{old}$, to get $M_1^{new} = arg\ Min_{M_1}\ KL(M_1, M_2^{old})$
**Step 2**   Fix $M_1 = M_1^{old}$, to get $M_2^{new} = arg\ Min_{M_2}\ KL(M_1^{old}, M_2)$

The ALTMIN iteration will finally converge to a local minimum of $K(M_1, M_2)$. We can have a similar procedure for $\max_{M_1,M_2} K(M_1, M_2)$ via replacing $Min$ by $Max$.

Since the above scheme bases on the two complement YING and YANG Bayesian representations and their Kullback divergence for their marital status, we call it *Bayesian-Kullback YING-YANG* learning scheme. Furthermore, under this scheme we call each obtained YING-YANG pair that is sensible for learning purpose as a *Bayesian-Kullback YING-YANG Machine* or *YING-YANG* machine shortly.

## 3   UNIFIED EXISTING LEARNINGS

Let $P_{M_1}(x) = P_0(x)$ by eq.(1) and put it into eq.(2), through certain mathematics we can get $K(M_1, M_2) = h_{M_1} - h_{\alpha M_1} - q_{M_{1,2}} + D$ with $D$ independent of $M_1, M_2$ and $h_{M_1}, h_{\alpha M_1}, q_{M_{1,2}}$ given by Eqs.(E1)(E2)&(E4) in Tab.2 respectively. The larger

is the $h_{M_1}$, the more discriminative or separable are the representations in $Y$ for the input data set. The larger is the $h_{\alpha M_1}$, the more concentrated the representations in $Y$. The larger is the $q_{M_{1,2}}$, the better $P_{M_2}(x|y)$ fits the input data.

Therefore, $\min_{M_1,M_2} K(M_1, M_2)$ consists of (1) best fitting of $P_{M_2}(x|y)$ on input data via $\max q_{M_{1,2}}$, which is desirable, (2) producing more concentrated representations in $Y$ to occupy less resource, which is also desirable and is the behind reason for solving the problem of selecting cluster number in clustering analysis Xu(1995a&c, 1996a), (3) but with the cost of less discriminative representations in $Y$ for the input data. Inversely, $\max_{M_1,M_2} K(M_1, M_2)$ consists of (1) producing best discriminative or separable representation $P_{M_1}(y|x)$ in $Y$ for the input data set, which is desirable, in the cost of (2) producing a more uniform representation in $Y$ to fully occupy the resource, and (3) causing $P_{M_2}(x|y)$ away from fitting input data.

Shown in Table 2 are the unified existing unsupervised learnings. For the case *H-f-W*, we have $h_{M_1} = h$, $h_{\alpha M_1} = h_\alpha$, $q_{M_{1,2}} = q_{M_2}$, and $\min_{M_1} K(M_1, M_2)$ results in $P_{M_2}(y) = P_{M_1}(y) = \alpha_y$ and $P_{M_2}(x|y)P_{M_2}(y) = P_{M_2}(x)P_{M_1}(y|x)$ with $P_{M_2}(x) = \sum_{y=1}^{k} P_{M_2}(x|y)P_{M_2}(y)$. In turn, we get $K(M_1, M_2) = -L_{M_2} + D$ with $L_{M_2}$ being the likelihood given by eq.(E5), i.e., we get maximum likelihood estimation on mixture model. In fact, the ALTMIN given in Tab.2 leads us to exactly the EM algorithm by Dempster et al(1977). Also, here $P_{M_1}(x,y)$, $P_{M_2}(x,y)$ is equivalent to the data submanifold $\mathcal{D}$ and model submanifold $\mathcal{M}$ in the *Information Geometry* theory (Amari, 1995a&b), with the ALTMIN being the *em* algorithm. As shown in Xu(95a), the cases also includes the MDL auto-encoder (Hinton & Zemel, 1994) and Multi-sets modeling (Xu, 1995e).

For the case *Single-M*, the $h_{M_1} - h_{\alpha M_1}$ is actually the information transmitted by the YANG part from $x$ to $y$. In this case, its minimization produces a non-sensible model for learning. However, its maximization is exactly the Informax learning scheme (Linsker, 1989; Atick & Redlich, 1990; Bell & Sejnowski, 1995). Here, we clear up a confusion made in Xu(95a&b) where the minimization was mistakenly considered.

For the case *H-m-W*, the $h_{M_1} - h_{\alpha M_1} - q_{M_{1,2}}$ is just the $-F(d; \theta, Q)$ used by Dayan et al (1995) and Hinton et al (1995) for Helmholtz machine. We can set up the detailed correspondence that (i) here $P_{M_1}(y|x_i)$ is their $Q_\alpha$; (ii) $\log P_{M_2}(x,y$ is their $-E_\alpha$; and (iii) their $P_\alpha$ is $P_{M_2}(y|x) = P_{M_2}(x|y)P_{M_2}(y) / \sum_y P_{M_2}(x|y)P_{M_2}(y)$. So, we get a new perspective for Helmholtz machine. Moreover, we know that $K(M_1, M_2)$ becomes a negative likelihood only when $P_{M_2}(x|y)P_{M_2}(y) = P_{M_2}(x)P_{M_1}(y|x)$, which is usually not true when the YANG and YING parts are both parametric. So Helmholtz machine is not equivalent to maximum likelihood learning in general with a gap depending on $P_{M_2}(x|y)P_{M_2}(y) - P_{M_2}(x)P_{M_1}(y|x)$. The equivalence is approximately acceptable only when the family of $P_{M_2}(x|y)$ or/and $P_{M_1}(y|x_i)$ is large enough or $M_2$, $M_1$ are both linear with gaussian density.

In Tab.4, the case *Single-M* under $K(M_2, M_1)$ is the classical maximum likelihood (ML) learning for supervised learning which includes the least square learning by back propagation (BP) for feedfarward net as a special case. Moreover, its counterpart for a backward net as inverse mapping is the case *Single-F* under $K(M_1, M_2)$.

## 4  NEW LEARNING MODELS

**First**, a number of variants for the above existing models are given in Table 2.

**Second**, a particular new model can be obtained from the case *H-m-W* by changing $\min_{M_1,M_2}$ into $\max_{M_1,M_2}$. That is, we have $\max_{M_1,M_2} [h_{M_1} - h_{\alpha M_1} - q_{M_{1,2}}]$, shortly

**Table 2: BKC-YY Machine for Unsupervised Learning ( Part I) :** $K(M_1, M_2)$
Given Data $\{x_i\}_{i=1}^N$, Fix $P_{M_1}(x) = P_0(x)$ by eq.(1), and thus $K(M_1, M_2) = K_b + D$, with $D$ irrelevant to $M_1, M_2$ and $K_b$ given by the following formulae and table:

$$h = \frac{1}{N} \sum_{i,y}^{N,k} P(y|x_i) \log P(y|x_i), \qquad h_{M_1} = \frac{1}{N} \sum_{i,y} P_{M_1}(y|x_i) \log P_{M_1}(y|x_i), \qquad (E1)$$

$$h_{\alpha M_1} = \sum_y \alpha_y^{M_1} \log \alpha_y^{M_1}, \qquad \alpha_y^{M_1} = \frac{1}{N} \sum_i P_{M_1}(y|x_i), \qquad h_\alpha = \sum_y \alpha_y \log \alpha_y, \qquad (E2)$$

$$\alpha_y = \frac{1}{N} \sum_i P(y|x_i), \qquad P(y|x_i) = \alpha_y P_{M_2}(x_i|y) / \sum_y \alpha_y P_{M_2}(x_i|y), \qquad (E3)$$

$$q_{M_{1,2}} = \frac{1}{N} \sum_{i,y} P_{M_1}(y|x_i) \log P_{M_2}(x_i|y), \qquad q_{M_2} = \frac{1}{N} \sum_{i,y} P(y|x_i) \log P_{M_2}(x_i|y), \qquad (E4)$$

$$L_{M_2}^\alpha = \frac{1}{N} \sum_{i,y} \alpha_y \log P_{M_2}(x_i|y), \qquad L_{M_2} = \frac{1}{N} \sum_i \log \sum_y \alpha_y P_{M_2}(x_i|y) \qquad (E5)$$

| Marriage Status | H-f-W | Single-M | Single-F | H-m-W | W-f-H |
|---|---|---|---|---|---|
| Condition | $P_{M_1}(y|x)$ free, i.e., $P_{M_1}(y|x)$ $= P(y|x)$ | $P_{M_2}(y)$ $= P_{M_1}(y)$ $P_{M_2}(x|y)$ $= P_{M_2}(x)$ $= P_0(x)$ | $P_{M_1}(y|x)$ $= P_{M_1}(y)$ | $P_{M_1}(y|x)$ and $P_{M_2}(x|y)$ | Uniform $P_{M_2}(y)$, and free $P_{M_2}(x|y)$ $= P(x|y)$ |
| $K_b$ | $h - h_\alpha - q_{M_2}$ $= -L_{M_2}$ (min) | $h_{M_1} - h_{\alpha M_1}$ (max) | $-L_{M_2}^\alpha$ (min) | $[h_{M_1} - h_{\alpha M_1}$ $- q_{M_{1,2}}]$ (min) | $h_{\alpha M_1}$ (min) |
| ALTMIN | S1: Fix $M_2$, get $P(y|x_i)$ $\alpha_y$ by (E3), $\alpha_y^{M_1}$ by (E2) S2: get $M_2$ by max $q_{M_2}$. | Get $M_1$ by max $h_{M_1} - h_{\alpha M_1}$ | Get $M_2$ by max $L_{M_2}^\alpha$. | S1: Fix $M_2$, get $M_1$ by min $[h_{M_1}$ $- h_{\alpha M_1} - q_{M_{1,2}}]$ S2: Fix $M_1$, get $M_2$ by max $q_{M_{1,2}}$. | Get $M_1$ by min $h_{\alpha M_1}$ |
| | Repeat **S1, S2.** | No Repeat | No Repeat | Repeat **S1, S2.** | No Repeat |
| Existing Equiv--lent models | 1. ML on Mixtures & EM (Dem77) 2. Information geometry (Amari95) 3. MDL Auto-encoder (Hin94) 4. Multi-sets modeling (Xu94,95) | Informax, Maximum mutual Information (Lin89) (Ati90) (Bel95) | Dupli-cated models by ML learning on input data. | Helm-holtz machine (Hin95) (Day95) | Related to PCA |
| New Results | 1. For H-f-W type , we have: Three VQ variants when $P_{M_2}(x|y)$ is Gaussian. Also, criteria for selecting the correct $k$ for VQ or clustering (Xu95a&c). 2. For H-m-W type, we have: Robust PCA + criterion for determining subspace dimension (Xu, 95c). | | | | |
| Variants | 1. More smooth $P_{M_1}(x)$ given by Parzen window estimate. 2. Factorial coding $P_{M_2}(y) = \prod_i P_{M_2}(y_i)$ with binary $y = [y_1 \cdots, y_m]$. 3. Factorial coding $P_{M_1}(y|x) = \prod_i P_{M_2}(y_i|x)$ with binary $[y_1 \cdots, y_m]$. 4. Replace '$\sum_y \cdot$' in all the above items by '$\int_y \cdot dy$' for real $y$. | | | | |

Note: H–Husband, W–Wife, f–follows, M–Male, F–Female, m–matches. X-f-Y stands for X part is free. Single-X stands for the other part broken. H-m-W stands for both parts being parametric. '(min)' stands for min $K_b$ and '(max)' stands for max $K_b$.

**Table 3: BKC-YY Machine for Unsupervised Learning ( Part II) :** $K(M_2, M_1)$
Given Data $\{x_i\}_{i=1}^{N}$, Fix $P_{M_1}(x) = P_0(x)$ by eq.(1), and thus $K(M_2, M_1) = K_b + D$, with $D$ irrelevant to $M_1, M_2$ and $K_b$ given by the following formulae and table:

$$h_{M_2} = \frac{1}{N} \sum_i P_{M_2}(x_i) \log P_{M_2}(x_i), \quad P_{M_2}(x) = \sum_y P_{M_2}(x|y)P_{M_2}(y), \tag{E6}$$

$$h_{\alpha M_2} = \sum_y \alpha_y^{M_2} \log \alpha_y^{M_2}, \quad \alpha_y^{M_2} = \frac{[\prod_i P_{M_1}(y|x_i)]^{1/N}}{\sum_y [\prod_i P_{M_1}(y|x_i)]^{1/N}}, \quad L_{\alpha M_1} = \sum_y \log \alpha_y^{M_1} \tag{E7}$$

$$L_{M_{1,2}} = \frac{1}{N} \sum_{i,y} \alpha_y^{M_2} \log P_{M_1}(y|x_i), \quad L_{M_1} = \frac{1}{N} \sum_{i,y} \alpha_y^{M_1} \log P_{M_1}(y|x_i), \tag{E8}$$

$$h_{M_2}^{\alpha} = \frac{1}{N} \sum_{i,y} \alpha_y P_{M_2}(x_i|y) \log P_{M_2}(x_i|y), \tag{E9}$$

$$q_{M_{2,1}} = \frac{1}{N} \sum_{i,y} \alpha_y^{M_1} P_{M_2}(x_i|y) \log P_{M_1}(y|x_i), \tag{E10}$$

$$(\alpha_y^{M_1}, h_{\alpha M_1}, L_{M_1} \text{ given in Table 1})$$

| Marriage Status | H-f-W | Single-M | Single-F | H-m-W | W-f-H |
|---|---|---|---|---|---|
| Condition | The same as those in Table 1. | | | | |
| $K_b$ | $h_{M_2}$ <br><br><br><br><br>(max) | $[h_{\alpha M_2} -$ <br> $-L_{M_{1,2}}]$ <br><br><br><br>(min) | $[h_{\alpha M_1} -$ <br> $- L_{M_1}]$ <br> (if forcing $P_{M_1}(y) =$ $P_{M_2}(y)$) <br> (min) | $h_{M_2}^{\alpha}$ <br><br><br><br><br>(max) | $[h_{M_2}^{\alpha} +$ <br> $+ h_{\alpha M_1} -$ <br> $-q_{M_{2,1}}]$ <br><br><br>(min) | $-L_{\alpha M_1}$ <br><br><br><br><br>(min) |
| ALTMIN | Get $M_2$ by *max* $h_{M_2}$ | **S1:** Fix $M_1$, get $\alpha_y^{M_2}$ by (E7). <br> **S2:** update $M_1$ by max $L_{M_{1,2}}$ | **S1:** Fix $M_1$, get $\alpha_y^{M_1}$ by (E2). in Tab.1 <br> **S2:** update $M_1$ by max $L_{M_1}$ | Get $M_2$ by max $h_{M_2}^{\alpha}$. | **S1:** Fix $M_2$, get $M_1$ by min $[h_{\alpha M_1}$ $-q_{M_{2,1}}]$ <br> **S2:** Fix $M_1$, get $M_2$ by min $h_{M_2}^{\alpha} - q_{M_{2,1}}$ | Get $M_1$ by max $L_{\alpha M_1}$ |
| | No Repeat | Repeat **S1, S2** | Repeat **S1, S2** | No Repeat | Repeat **S1, S2** | No Repeat |
| Existing models | no new ! | no new ! | no new ! | no new ! | no new ! | no new ! |
| Variants | Similar to those in Table 1. | | | | | |

**Table 4: BKC-YY Machine for Supervised Learning**
Given Data $\{x_i, y_i\}_{i=1}^{N}$, Fix $P_{M_1}(x) = P_0(x)$ by eq.(1).

$$h_{M_1}^s = \frac{1}{N} \sum_i P_{M_1}(y_i|x_i) \log P_{M_1}(y_i|x_i), \quad h_{M_2}^s = \frac{1}{N} \sum_i P_{M_2}(x_i|y_i) \log P_{M_2}(x_i|y_i), \tag{E11}$$

$$q_{M_{1,2}}^s = \frac{1}{N} \sum_i P_{M_1}(y_i|x_i) \log P_{M_2}(x_i|y_i), \quad q_{M_{2,1}}^s = \frac{1}{N} \sum_i P_{M_2}(x_i|y_i) \log P_{M_1}(y_i|x_i), \tag{E12}$$

$$L_{M_1}^s = \frac{1}{N} \sum_i \log P_{M_1}(y_i|x_i), \quad L_{M_2}^s = \frac{1}{N} \sum_i \log P_{M_2}(x_i|y_i), \tag{E13}$$

| Marriage Status | $K(M_1, M_2) = K_b + D$ | | | $K(M_2, M_1) = K_b + D$ | | |
|---|---|---|---|---|---|---|
| | Single-M | Single-F | H-m-W | Single-M | Single-F | H-m-W |
| $K_b$ | $h_{M_1}^s$ (max) | $-L_{M_2}^s$ (min) | $h_{M_1}^s - q_{M_{1,2}}^s$ (min) | $-L_{M_1}^s$ (min) | $h_{M_2}^s$ (max) | $h_{M_2}^s - q_{M_{2,1}}^s$ (min) |
| Feature | minimum entropy(ME) F-net | ML <br> B-net | Mixed F-B net | ML <br> F-net | minimum entropy B-net | Mixed B-F net |
| Existing models | no new ! | BP on B-net | no new ! | BP on F-net | no new ! | no new ! |

denoted by *H-m-W-Max*. This model is a dual to the Helmholtz machine in order to focus on getting best discriminative or separable representations $P_{M_1}(y|x)$ in $Y$ instead of best fitting of $P_{M_2}(x|y)$ on input data.

**Third**, by replacing $K(M_1, M_2)$ with $K(M_2, M_1)$, in Table 3 we can obtain new models that are the counterparts of those given in Table 2. For the case *H-f-W*, its $\max_{M_1,M_2}$ gives minimum entropy estimate on $P_{M_2}(x)$ instead of maximum likelihood estimate on $P_{M_2}(x)$ in Table 2. For the case *Single-M*, it will function similarly to the case *Single-F* in Table 2, but with minimum entropy on $P_{M_1}(y|x)$ in Table 2 replaced by maximum likelihood on $P_{M_1}(y|x)$ here. For the case *H-m-W*, the focus shifts from on getting best fitting of $P_{M_2}(x|y)$ on input data to on getting best discriminative representations $P_{M_1}(y|x)$ in $Y$, which is similar to the just mentioned *H-m-W-Max*, but with minimum entropy on $P_{M_1}(y|x)$ replaced by maximum likelihood on $P_{M_1}(y|x)$. The other two cases in Table 3 have been also changed similarly from those in Table 2.

**Fourth**, several new model have also been proposed in Table 4 for supervised learning. Instead of maximum likelihood, the new models suggest learning by minimum entropy or a mix of maximum likelihood and minimum entropy.

**Finally**, further studies on the other status in Table 1 are needed. Heuristically, we can also treat the case *H-m-W* by two separated steps. We first get $M_1$ by $\max[h_{M_1} - h_{\alpha M_1}]$, and then get $M_2$ by $\max q_{M_{1,2}}$; or we first get $M_2$ by $\min[h - h_\alpha - q_{M_2}]$ and then get $M_1$ by $\min[h_{M_1} - h_{\alpha M_1} - q_{M_{1,2}}]$. The two algorithms attempt to get both a good discriminative representation by $P_{M_1}(y|x)$ and a good fitting of $P_{M_2}(x|y)$ on input data. However whether they work well needs to be tested experimentally.

We are currently conducting experiments on comparison several of the above new models against their existing counterparts.

**Acknowledgements** *The work was Supported by the HK RGC Earmarked Grant CUHK250/94E.*

## References

Amari, S(1995a) [Amari95] " Information geometry of the EM and em algorithms for neural networks", *Neural Networks 8*, to appear.
Amari, S(1995b), *Neural Computation 7*, pp13-18.
Atick, J.J. & Redlich, A.N. (1990) [Ati90], *Neural Computation* Vol.2, No.3, pp308-320.
Bell A. J. & Sejnowski, T. J.(1995) [Bel95], *Neural Computation* Vol.7, No.6, 1129-1159.
Byrne, W. (1992), *IEEE Trans. Neural Networks 3*, pp612-620.
Csiszar, I., (1975), *Annals of Probability 3*, pp146-158.
Dayan, P., Hinton, G. E., & Neal, R. N. (1995) [Day95], *Neural Computation* Vol.7, No.5, 889-904.
Dempster, A.P., Laird, N.M., & Rubin, D.B. (1977) [Dem77], *J. Royal Statist. Society, B39*, 1-38.
Hathaway, R.J.(1986), *Statistics & Probability Letters 4*, pp53-56.
Hinton, G. E., et al, (1995) [Hin95], *Science 268*, pp1158-1160.
Hinton, G. E. & Zemel, R.S. (1994) [Hin94], *Advances in NIPS 6*, pp3-10.
Linsker, R. (1989) [Lin89], *Advances in NIPS 1*, pp186-194.
Neal, R. N.& Hinton, G. E(1993), *A new view of the EM algorithm that justifies incremental and other variants*, preprint.
Xu, L. (1996), "How Many Clusters ? : A YING-YANG Machine Based Theory For A Classical Open Problem In Pattern Recognition", to appear on *Proc. IEEE ICNN96*.
Xu, L. (1995a), "YING-YANG Machine: a Bayesian-Kullback scheme for unified learnings and new results on vector quantization", Keynote talk, Proc. Intl Conf. on Neural Information Processing (ICONIP95), Oct 30 - Nov. 3, 1995, pp977-988.
Xu, L.(1995b), "YING-YANG Machine for Temporal Signals", Keynote talk, Proc IEEE intl Conf. Neural Networks & Signal Processing, Vol.I, pp644-651, Nanjing, 10-13, 1995.
Xu, L. (1995c), "New Advances on The YING-YANG Machine", Invited paper, Proc. of 1995 Intl. Symposium on Artificial Neural Networks, ppIS07-12, Dec. 18-20, Taiwan.
Xu, L. (1995d), "Cluster Number Selection, Adaptive EM Algorithms and Competitive Learnings", Invited paper, Proc. Intl Conf. on Neural Information Processing (ICONIP95), Oct 30 - Nov. 3, 1995, Vol.II, pp1499-1502.
Xu, L. (1995e), Invited paper, Proc. WCNN95, Vol.I, pp35-42. Also, Invited paper, Proc. IEEE ICNN 1994, ppI315-320.
Xu, L., & Jordan, M.I. (1993). *Proc. of WCNN'93*, Portland, OR, Vol. II, 431-434.